# Tighter PAC-Bayes Bounds

**Amiran Ambroladze**
Dep. of Mathematics
Lund University/LTH
Box 118, S-221 00 Lund, SWEDEN
amiran.ambroladze@math.lth.se

**Emilio Parrado-Hernández**
Dep. of Signal Processing and Communications
University Carlos III of Madrid
Leganés, 28911, SPAIN
emipar@tsc.uc3m.es

**John Shawe-Taylor**
Dep. of Computer Science
University College London
Gower Street,
London WC1E 6BT, UK
jst@cs.ucl.ac.uk

## Abstract

This paper proposes a PAC-Bayes bound to measure the performance of Support Vector Machine (SVM) classifiers. The bound is based on learning a prior over the distribution of classifiers with a part of the training samples. Experimental work shows that this bound is tighter than the original PAC-Bayes, resulting in an enhancement of the predictive capabilities of the PAC-Bayes bound. In addition, it is shown that the use of this bound as a means to estimate the hyperparameters of the classifier compares favourably with cross validation in terms of accuracy of the model, while saving a lot of computational burden.

## 1 Introduction

Support vector machines (SVM) implement linear classifiers in a high-dimensional feature space using the kernel trick to enable a dual representation and efficient computation.

The danger of overfitting in such high-dimensional spaces is countered by maximising the margin of the classifier on the training examples. For this reason there has been considerable interest in bounds on the generalisation in terms of the margin.

Early bounds have relied on covering number computations [7], while later bounds have considered Rademacher complexity. The tightest bounds for practical applications appear to be the PAC-Bayes bound [4, 5]. In particular the form given in [3] is specially attractive for margin classifiers, like SVM. The PAC-Bayesian bounds are also present in other Machine Learning models such as Gaussian Processes [6].

The aim of this paper is to consider a refinement of the PAC-Bayes approach and investigate whether it can improve on the original PAC-Bayes bound and uphold its capabilities of delivering reliable model selection.

The standard PAC-Bayes bound uses a Gaussian prior centred at the origin in weight space. The key to the new bound is to use part of the training set to compute a more informative prior and then compute the bound on the remainder of the examples relative to this prior. The bounds are tested experimentally in several classification tasks, including the model selection, on common benchmark datasets.

The rest of the document is organised as follows. Section 2 briefly reviews the PAC-Bayes bound for SVMs obtained in [3]. The new bound obtained by means of the refinement of the prior is presented in Section 3. The experimental work, included in Section 4, compares the tightness of the new bound with the original one and indicates about its usability in a model selection task. Finally, the main conclusions of this work are outlined in Section 5.

## 2 PAC-Bayes Bound

This section is devoted to a brief review of the PAC-Bayes Bound Theorem of [3]. Let us consider a distribution $\mathcal{D}$ of patterns $\mathbf{x}$ lying in a certain input space $\mathcal{X}$, with their corresponding output labels $y$, $y \in \{-1, 1\}$. In addition, let us also consider a distribution $Q$ over the classifiers $c$. For every classifier $c$, the following two error measures are defined:

**Definition** *(True error)* The true error $c_{\mathcal{D}}$ of a classifier $c$ is defined as the probability of misclassifying a pair pattern-label $(\mathbf{x}, y)$ selected at random from $\mathcal{D}$

$$c_{\mathcal{D}} \equiv \Pr_{(\mathbf{x},y)\sim\mathcal{D}}(c(\mathbf{x}) \neq y)$$

**Definition** *(Empirical error)* The empirical error $\hat{c}_S$ of a classifier $c$ on a sample $S$ of size $m$ is defined as the rate of errors on a set $S$

$$\hat{c}_S \equiv \Pr_{(\mathbf{x},y)\sim S}(c(\mathbf{x}) \neq y) = \frac{1}{m}\sum_{i=1}^{m} I(c(\mathbf{x}_i) \neq y_i)$$

where $I(\cdot)$ is a function equal to 1 if the argument is true and equal to 0 if the argument is false.

Now we can define two error measures on the distribution of classifiers: the true error, $Q_{\mathcal{D}} \equiv \mathbb{E}_{c\sim Q}c_{\mathcal{D}}$, as the probability of misclassifying an instance $\mathbf{x}$ chosen from $\mathcal{D}$ with a classifier $c$ chosen according to $Q$; and the empirical error $\hat{Q}_S \equiv \mathbb{E}_{c\sim Q}\hat{c}_S$, as the probability of classifier $c$ chosen according to $Q$ misclassifying an instance $\mathbf{x}$ chosen from a sample $S$.

For these two quantities we can derive the PAC-Bayes Bound on the true error of the distribution of classifiers:

**Theorem 2.1** (PAC-Bayes Bound) *For all prior distributions $P(c)$ over the classifiers $c$, and for any $\delta \in (0, 1]$*

$$Pr_{S\sim\mathcal{D}^m}\left(\forall Q(c) : KL(\hat{Q}_S||Q_{\mathcal{D}}) \leq \frac{KL(Q(c)||P(c)) + \ln(\frac{m+1}{\delta})}{m}\right) \geq 1 - \delta,$$

*where $KL$ is the Kullback-Leibler divergence, $KL(p||q) = q\ln\frac{q}{p} + (1-q)\ln\frac{1-q}{1-p}$ and $KL(Q(c)||P(c)) = \mathbb{E}_{c\sim Q}\ln\frac{Q(c)}{P(c)}$.*

The proof of the theorem can be found in [3].

This bound can be particularised for the case of linear classifiers in the following way. The $m$ training patterns define a linear classifier that can be represented by the following equation[1]:

$$c(\mathbf{x}) = \text{sign}(\mathbf{w}^T\phi(\mathbf{x})) \tag{1}$$

where $\phi(\mathbf{x})$ is a nonlinear projection to a certain feature space where a linear classification actually takes place, and $\mathbf{w}$ is a vector from that feature space that determines the separating plane.

For any vector $\mathbf{w}$ we can define a stochastic classifier in the following way: we choose the distribution $Q = Q(\mathbf{w}, \mu)$ to be a spherical Gaussian with identity covariance matrix centred on the direction given by $\mathbf{w}$ at a distance $\mu$ from the origin. Moreover, we can choose the prior $P(c)$ to be a spherical Gaussian with identity covariance matrix centred on the origin. Then, for classifiers of the form in equation (1) performance can be bounded by

**Corollary 2.2** (PAC-Bayes Bound for margin classifiers [3]) *For all distributions $\mathcal{D}$, for all classifiers given by $\mathbf{w}$ and $\mu > 0$, for all $\delta \in (0, 1]$, we have*

$$Pr\left( KL(\hat{Q}_S(\mathbf{w}, \mu) || Q_{\mathcal{D}}(\mathbf{w}, \mu)) \leq \frac{\frac{\mu^2}{2} + \ln(\frac{m+1}{\delta})}{m} \right) \geq 1 - \delta.$$

It can be shown (see [3]) that

$$\hat{Q}_S(\mathbf{w}, \mu) = \mathbb{E}_m[\tilde{F}(\mu\gamma(\mathbf{x}, y))] \tag{2}$$

where $\mathbb{E}_m$ is the average over the $m$ training examples, $\gamma(\mathbf{x}, y)$ is the normalised margin of the training patterns

$$\gamma(\mathbf{x}, y) = \frac{y\mathbf{w}^T\phi(\mathbf{x})}{\|\phi(\mathbf{x})\|\|\mathbf{w}\|} \tag{3}$$

and $\tilde{F} = 1 - F$, where $F$ is the cumulative normal distribution

$$F(x) = \int_{-\infty}^{x} \frac{1}{\sqrt{2\pi}} e^{-x^2/2} \mathrm{d}x. \tag{4}$$

Note that the SVM is a thresholded linear classifier expressed as (1) computed by means of the kernel trick [2]. The generalisation error of such a classifier can be bounded by at most twice the true (stochastic) error $Q_{\mathcal{D}}(\mathbf{w}, \mu)$ in Corollary 2.2, (see [4]);

$$\Pr_{(\mathbf{x}, y) \sim \mathcal{D}} \left( \mathrm{sign}(\mathbf{w}^T\phi(\mathbf{x})) \neq y \right) \leq 2Q_{\mathcal{D}}(\mathbf{w}, \mu)$$

for all $\mu$.

## 3 Choosing a prior for the PAC-Bayes Bound

Our first contribution is motivated by the fact that the PAC-Bayes bound allows us to choose the prior distribution, $P(c)$. In the standard application of the bound this is chosen to be a Gaussian centred at the origin. We now consider learning a different prior based on training an SVM on a subset $R$ of the training set comprising $r$ training patterns and labels. In the experiments this is taken as a random subset but for simplicity of the presentation we will assume these to be the last $r$ examples $\{\mathbf{x}_k, y_k\}_{k=m-r+1}^{m}$ in the description below.

With these $r$ examples we can determine an SVM classifier, $\mathbf{w}_r$ and form a prior $P(\mathbf{w}|\mathbf{w}_r)$ consisting of a Gaussian distribution with identity covariance matrix centred on $\mathbf{w}_r$.

The introduction of this prior $P(\mathbf{w}|\mathbf{w}_r)$ in Theorem 2.1 results in the following new bound.

**Corollary 3.1** (Single Prior based PAC-Bayes Bound for margin classifiers) *Let us consider a prior on the distribution of classifiers consisting in a spherical Gaussian with identity covariance centred along the direction given by $\mathbf{w}_r$ at a distance $\eta$ from the origin. Then, for all distributions $D$, for all classifiers $\mathbf{w}_m$ and $\mu > 0$, for all $\delta \in (0, 1]$, we have*

$$Pr_{S \sim D}\left( KL(\hat{Q}_{S \backslash R}(\mathbf{w}_m, \mu) || Q_D(\mathbf{w}_m, \mu)) \leq \frac{\frac{\|\eta\mathbf{w}_r - \mu\mathbf{w}_m\|^2}{2} + \ln(\frac{m-r+1}{\delta})}{m - r} \right) \geq 1 - \delta$$

*where $\hat{Q}_{S \backslash R}$ is a stochastic measure of the error of the classifier on the $m - r$ samples not used to learn the prior. This stochastic error is computed as indicated in equation (2) averaged over $S \backslash R$.*

**Proof** Since we separate $r$ instances to learn the prior, the actual size of the training set to which we apply the bound is $m - r$. In addition, the stochastic error must be computed only on the instances not used to learn the prior, i.e. the subset $S \backslash R$.

The KL divergence between prior and posterior is computed as follows:

$$
\begin{aligned}
\mathrm{KL}(Q(\mathbf{w}) || P(\mathbf{w})) &= \mathbb{E}_{\mathbf{w} \sim Q} \ln \frac{Q(\mathbf{w})}{P(\mathbf{w})} \\
&= \mathbb{E}_{\mathbf{w} \sim Q} \ln \frac{\exp\left(-\frac{1}{2}(\mathbf{w} - \mu\mathbf{w}_m)^T(\mathbf{w} - \mu\mathbf{w}_m)\right)}{\exp\left(-\frac{1}{2}(\mathbf{w} - \eta\mathbf{w}_r)^T(\mathbf{w} - \eta\mathbf{w}_r)\right)} \\
&= \mathbb{E}_{\mathbf{w} \sim Q}\left[-\frac{1}{2}(\mathbf{w} - \mu\mathbf{w}_m)^T(\mathbf{w} - \mu\mathbf{w}_m) + \frac{1}{2}(\mathbf{w} - \eta\mathbf{w}_r)^T(\mathbf{w} - \eta\mathbf{w}_r)\right] \\
&= \mathbb{E}_{\mathbf{w} \sim Q}\left(\mu\mathbf{w}_m^T\mathbf{w}\right) - \frac{1}{2}\mu^2\mathbf{w}_m^T\mathbf{w}_m - \mathbb{E}_{\mathbf{w} \sim Q}\left(\eta\mathbf{w}^T\mathbf{w}_r\right) + \frac{1}{2}\eta^2\mathbf{w}_r^T\mathbf{w}_r
\end{aligned}
$$

Taking expectations using $\mathbb{E}_{\mathbf{w} \sim Q} \mathbf{w} = \mu \mathbf{w}_m$ we arrive at

$$\frac{1}{2} ||\mu \mathbf{w}_m - \eta \mathbf{w}_r||^2$$

∎

Intuitively, if the selection of the prior is appropriate, the bound can be tighter than the one given in Corollary 2.2 when applied to the SVM weight vector on the whole training set. It is perhaps worth stressing that the bound holds for all $\mathbf{w}_m$ and so can be applied to the SVM trained on the whole set. This might at first appear as 'cheating', but the critical point is that the bound is evaluated on the set $S \backslash R$ not involved in generating the prior. The experimental work illustrates how in fact this bound can be tighter than the standard PAC-Bayes bound.

Moreover, the selection of the prior may be further refined in exchange for a very small increase in the penalty term. This can be achieved with the application of the following result.

**Theorem 3.2** (Bound for several priors) *Let $\{P_j(c)\}_{j=1}^J$ be a set of possible priors that can be selected with positive weights $\{\pi_j\}_{j=1}^J$ so that $\sum_{j=1}^J \pi_j = 1$. Then, for all priors $P(c) \in \{P_j(c)\}_{j=1}^J$, for all posterior distributions $Q(c)$, for all $\delta \in (0, 1]$,*

$$Pr_{S \sim \mathcal{D}^m} \left( \forall Q(c), \forall j : KL(\hat{Q}_S || Q_\mathcal{D}) \leq \frac{KL(Q(c)||P_j(c)) + \ln \frac{m+1}{\delta} + \ln \frac{1}{\pi_j}}{m} \right) \geq 1 - \delta,$$

**Proof** The bound in Theorem 2.1 can be particularised for a certain $P_j(c)$ with associated weight $\pi_j$ and with confidence $\delta \pi_j$

$$Pr_{S \sim \mathcal{D}^m} \left( \forall Q(c) : KL(\hat{Q}_S || Q_\mathcal{D}) > \frac{KL(Q(c)||P_j(c)) + \ln(\frac{m+1}{\delta \pi_j})}{m} \right) < \delta \pi_j,$$

Now let us combine the bounds for all the priors $\{P_j(c)\}_{j=1}^J$ with the union operation (we use the fact that $P(a \cup b) \leq P(a) + P(b)$).

$$Pr_{S \sim \mathcal{D}^m} \left( \begin{array}{l} \forall Q(c), \exists P(c) \in \{P_j(c)\}_{j=1}^J : \\ KL(\hat{Q}_S || Q_\mathcal{D}) > \frac{KL(Q(c)||P_j(c)) + \ln \frac{m+1}{\delta} + \ln \frac{1}{\pi_j}}{m} \end{array} \right) < \delta, \quad (5)$$

Finally, let us take the negation of (5) to arrive at the final result. ∎

This result can be also particularised for the case of SVM classifiers. The set of priors is constructed by allocating Gaussian distributions with identity covariance matrix along the direction given by $\mathbf{w}_r$ at distances $\{\eta_j\}_{j=1}^J$ from the origin where $\{\eta_j\}_{j=1}^J$ are real numbers. In such a case, we obtain

**Corollary 3.3** (Multiple Prior PAC-Bayes Bound for linear classifiers) *Let us consider a set $\{P_j(\mathbf{w}|\mathbf{w}_r, \eta_j)\}_{j=1}^J$ of prior distributions of classifiers consisting in spherical Gaussian distributions with identity covariance matrix centred on $\eta_j \mathbf{w}_r$, where $\{\eta_j\}_{j=1}^J$ are real numbers. Then, for all distributions $\mathcal{D}$, for all classifiers $\mathbf{w}$, for all $\mu > 0$, for all $\delta \in (0, 1]$, we have*

$$Pr_{S \sim \mathcal{D}} \left( KL(\hat{Q}_{S \backslash R}(\mathbf{w}, \mu) || Q_\mathcal{D}(\mathbf{w}, \mu)) \leq \frac{\frac{||\eta_j \mathbf{w}_r - \mu \mathbf{w}||^2}{2} + \ln(\frac{m-r+1}{\delta}) + \ln J}{m - r} \right) \geq 1 - \delta$$

**Proof** The proof is straightforward, substituting $\pi_j = \frac{1}{J}$ for all $j$ in Theorem 3.2 and computing the KL divergence between prior and posterior as in the proof of Corollary 3.1. ∎

Note that the $\{\eta_j\}_{j=1}^J$ must be chosen before we actually compute the posterior. However, the bound holds for all $\mu$. Therefore, a linear search can be implemented for the value of $\mu$ that leads to the tightest bound. In the case of several priors, the search is repeated for every prior and the reported value of the bound is the tightest. In Section 4 we present experimental results comparing this new bound to the standard PAC-Bayes bound and using it to guide model selection.

## 4   Experiments

The tightness of the new bound is evaluated in a model selection and classification task using some UCI [1] datasets (see their description in terms of number of instances, input dimension and number of positive/negative examples in Table 1).

| Problem | # samples | input dim. | Pos/Neg |
|---------|-----------|------------|---------|
| Wdbc | 569 | 30 | 357 / 212 |
| Image | 2310 | 18 | 1320 / 990 |
| Waveform | 5000 | 21 | 1647 / 3353 |
| Ringnorm | 7400 | 20 | 3664 / 3736 |

Table 1: Description of the datasets: for every set we give the number of patterns, number of input variables and number of positive/negative examples.

For every dataset, we obtain 50 different training/test set partitions with 80% of the samples forming the training set and the remaining 20% forming the test set.

With each of the partitions we learn a SVM classifier with Gaussian RBF kernel preceded by a model selection. The model selection consists in the determination of an optimal pair of hyperparameters $(C, \sigma)$. $C$ is the SVM trade-off between the maximisation of the margin and the minimisation of the hinge loss of the training samples, while $\sigma$ is the width of the Gaussian kernel. The best pair is sought in a $15 \times 15$ grid of parameters where $C \in \{0.02, 0.05, 0.1, 0.2, 0.5, 1, 2, 5, 10, 20, 50, 100, 200, 500, 1000\}$ and $\sigma \in \{\frac{1}{8}\sqrt{d}, \frac{1}{7}\sqrt{d}, \frac{1}{6}\sqrt{d}, \frac{1}{5}\sqrt{d}, \frac{1}{4}\sqrt{d}, \frac{1}{3}\sqrt{d}, \frac{1}{2}\sqrt{d}, \sqrt{d}, 2\sqrt{d}, 3\sqrt{d}, 4\sqrt{d}, 5\sqrt{d}, 6\sqrt{d}, 7\sqrt{d}, 8\sqrt{d}\}$, where $d$ is the input space dimension.

For completeness, this model selection is guided by the PAC-Bayes bound: we select the model corresponding to the pair that yields a lower value of $Q_D$ in the bound. Table 2 shows the value of the PAC-Bayes Bound averaged over the 50 training/test partitions. For every partition we use the minimum value of the bound resulting from all the pairs $(C, \sigma)$ of the grid. Note that this procedure is computationally less costly than the commonly used $N$-fold cross validation model selection, since it saves the training of $N$ classifiers (one for each fold) for each parameter combination.

| Problem | PAC-Bayes Bound | Test error rate |
|---------|-----------------|-----------------|
| Wdbc | $0.334 \pm 0.005$ | $0.073 \pm 0.021$ |
| Image | $0.254 \pm 0.003$ | $0.074 \pm 0.014$ |
| Waveform | $0.198 \pm 0.002$ | $0.089 \pm 0.008$ |
| Ringnorm | $0.212 \pm 0.002$ | $0.026 \pm 0.005$ |

Table 2: Averaged PAC-Bayes Bound and Test Error Rate obtained by the model that yielded the lowest bound in each of the 50 training/test partitions.

We repeated this experiment using the Prior PAC-Bayes Bound with different configurations for learning the prior distribution of classifiers. These configurations are defined by variations on the percentage of training patterns separated to compute the prior and on the number of scalings of the magnitude of that prior. The scalings represent different lengths $\eta$ of $||\mathbf{w}_r||$ equally spaced between $\eta = 1$ and $\eta = 100$. To summarize, for every training/test partition and for every pair (% patterns, # of scalings) we look at the pair $(C, \sigma)$ that outputs the smaller value of $Q_D$.

In this case, the use of the Prior PAC-Bayes Bound to perform the model selection increases the computational burden of using the PAC-Bayes one in the training of one classifier (the one used to learn the prior), in comparison to the extra $N$ classifiers needed by $N$-fold cross validation.

Table 3 displays both the average value and the sample standard deviation over the 50 realisations.

It seems that ten scalings of the prior are enough to obtain tighter bounds, since the use of 100 or 500 scalings does not improve the best results. With respect to the percentage of training instances left out to learn the prior, something close to 50% of the training set works well in the considered problems. It is worth mentioning that we treat each position in the Table as a separate experiment.

| Winsconsin Database of Breast Cancer (PAC-Bayes Bound = 0.334±0.005) | | | | | |
|---|---|---|---|---|---|
| Scalings | Percentage of training set used to compute the prior | | | | |
| | 10 % | 20% | 30% | 40% | 50% |
| 1 | $0.341 \pm 0.006$ | $0.351 \pm 0.007$ | $0.364 \pm 0.009$ | $0.379 \pm 0.011$ | $0.398 \pm 0.013$ |
| 10 | $0.337 \pm 0.010$ | $0.323 \pm 0.012$ | $0.314 \pm 0.012$ | $0.310 \pm 0.013$ | $\mathbf{0.306 \pm 0.018}$ |
| 100 | $0.319 \pm 0.007$ | $0.315 \pm 0.010$ | $\mathbf{0.313 \pm 0.011}$ | $0.315 \pm 0.013$ | $0.315 \pm 0.017$ |
| 500 | $0.324 \pm 0.007$ | $0.320 \pm 0.009$ | $\mathbf{0.319 \pm 0.011}$ | $0.321 \pm 0.013$ | $0.322 \pm 0.017$ |

| Image Segmentation (PAC-Bayes Bound = 0.254±0.003) | | | | | |
|---|---|---|---|---|---|
| Scalings | Percentage of training set used to compute the prior | | | | |
| | 10 % | 20% | 30% | 40% | 50% |
| 1 | $\mathbf{0.255 \pm 0.003}$ | $0.262 \pm 0.005$ | $0.274 \pm 0.003$ | $0.284 \pm 0.005$ | $0.300 \pm 0.008$ |
| 10 | $0.215 \pm 0.004$ | $0.203 \pm 0.006$ | $0.200 \pm 0.005$ | $0.188 \pm 0.007$ | $\mathbf{0.184 \pm 0.010}$ |
| 100 | $0.217 \pm 0.004$ | $0.203 \pm 0.007$ | $0.196 \pm 0.005$ | $0.187 \pm 0.007$ | $\mathbf{0.186 \pm 0.009}$ |
| 500 | $0.218 \pm 0.004$ | $0.204 \pm 0.007$ | $0.198 \pm 0.005$ | $0.189 \pm 0.007$ | $\mathbf{0.188 \pm 0.009}$ |

| Waveform (PAC-Bayes Bound = 0.198±0.002) | | | | | |
|---|---|---|---|---|---|
| Scalings | Percentage of training set used to compute the prior | | | | |
| | 10 % | 20% | 30% | 40% | 50% |
| 1 | $\mathbf{0.197 \pm 0.003}$ | $0.201 \pm 0.003$ | $0.207 \pm 0.003$ | $0.214 \pm 0.004$ | $0.222 \pm 0.005$ |
| 10 | $0.161 \pm 0.004$ | $0.156 \pm 0.004$ | $0.153 \pm 0.004$ | $\mathbf{0.150 \pm 0.005}$ | $0.151 \pm 0.005$ |
| 100 | $0.161 \pm 0.004$ | $0.155 \pm 0.004$ | $0.153 \pm 0.004$ | $\mathbf{0.152 \pm 0.005}$ | $0.153 \pm 0.005$ |
| 500 | $0.162 \pm 0.004$ | $0.157 \pm 0.004$ | $0.155 \pm 0.004$ | $\mathbf{0.154 \pm 0.005}$ | $0.155 \pm 0.005$ |

| Ringnorm (PAC-Bayes Bound = 0.212±0.002) | | | | | |
|---|---|---|---|---|---|
| Scalings | Percentage of training set used to compute the prior | | | | |
| | 10 % | 20% | 30% | 40% | 50% |
| 1 | $\mathbf{0.216 \pm 0.001}$ | $0.225 \pm 0.002$ | $0.236 \pm 0.002$ | $0.249 \pm 0.004$ | $0.265 \pm 0.002$ |
| 10 | $0.172 \pm 0.068$ | $0.140 \pm 0.047$ | $0.126 \pm 0.037$ | $0.116 \pm 0.030$ | $\mathbf{0.109 \pm 0.024}$ |
| 100 | $0.173 \pm 0.068$ | $0.139 \pm 0.047$ | $0.126 \pm 0.037$ | $0.117 \pm 0.030$ | $\mathbf{0.110 \pm 0.024}$ |
| 500 | $0.173 \pm 0.068$ | $0.140 \pm 0.047$ | $0.127 \pm 0.037$ | $0.117 \pm 0.030$ | $\mathbf{0.110 \pm 0.024}$ |

Table 3: Averaged Prior PAC-Bayes bound for different settings of percentage of training instances reserved to compute the prior and of number of scalings of the normalised prior.

However, one could have included the tuning of the pair (`% patterns, # of scalings`) in the model selection. This would have involved a further application of the union bound with the 20 entries of the Table for each problem, at the cost of adding an extra $\ln(20)/m$ (0.0053 for Wdbc and less for the other datasets) in the right part of Theorem 3.2. We decided to fix the number of scalings and the amount of training patterns to compute the prior since to perform all of the different options would augment the computational burden of the model selection.

In order to evaluate the predictive capabilities of the Prior PAC-Bayes bound as a means to select models with low test error rate, Table 4 displays the averaged test error corresponding to the models selected in the previous experiment (note that in this case the computational burden involved in determining the model is increased by the training of the SVM that learns the prior $\mathbf{w}_r$). Table 5 displays the test error rate obtained by SVMs with their hyperparameters tuned on the above mentioned grid by means of ten-fold cross-validation, that serves as a baseline method for comparison purposes.

According to the values shown in the tables, the Prior PAC-Bayes bound achieves tighter predictions of the generalization error of the randomized classifier in almost all cases.

Notice how the length of the prior is not so critical in comparison with its direction. The goodness of the latter relying on the subset of samples left out for the purpose of learning the prior classifier. Moreover it has to be remarked that this tightening of the bound does not appear to deliver any reduction in the capabilities to select a good model (such a case would imply that we can predict more accurately a bigger error rate, but our bound is able to predict accurately the same error rate as the PAC-Bayes Bound).

| | **Winsconsin Database of Breast Cancer** (PAC-Bayes Test Error = 0.073±0.021) | | | | |
|---|---|---|---|---|---|
| Scalings | Percentage of training set used to compute the prior | | | | |
| | **10 %** | **20%** | **30%** | **40%** | **50%** |
| **1** | **0.076 ± 0.020** | 0.076 ± 0.021 | 0.076 ± 0.021 | 0.076 ± 0.021 | 0.076 ± 0.021 |
| **10** | 0.075 ± 0.021 | 0.076 ± 0.021 | 0.075 ± 0.021 | 0.074 ± 0.021 | **0.072 ± 0.021** |
| **100** | 0.076 ± 0.021 | 0.076 ± 0.021 | 0.074 ± 0.021 | 0.074 ± 0.020 | **0.072 ± 0.021** |
| **500** | 0.076 ± 0.020 | 0.076 ± 0.021 | 0.074 ± 0.020 | 0.073 ± 0.020 | **0.072 ± 0.021** |

| | **Image Segmentation** (PAC-Bayes Test Error = 0.074±0.014) | | | | |
|---|---|---|---|---|---|
| Scalings | Percentage of training set used to compute the prior | | | | |
| | **10 %** | **20%** | **30%** | **40%** | **50%** |
| **1** | **0.078 ± 0.011** | 0.078 ± 0.011 | 0.078 ± 0.011 | 0.083 ± 0.019 | 0.100 ± 0.019 |
| **10** | 0.064 ± 0.011 | 0.066 ± 0.011 | 0.063 ± 0.014 | **0.054 ± 0.010** | 0.056 ± 0.011 |
| **100** | 0.064 ± 0.011 | 0.063 ± 0.011 | 0.061 ± 0.011 | 0.059 ± 0.011 | **0.057 ± 0.012** |
| **500** | 0.064 ± 0.011 | 0.063 ± 0.011 | 0.061 ± 0.011 | 0.059 ± 0.011 | **0.057 ± 0.012** |

| | **Waveform** (PAC-Bayes Test Error = 0.089±0.008) | | | | |
|---|---|---|---|---|---|
| Scalings | Percentage of training set used to compute the prior | | | | |
| | **10 %** | **20%** | **30%** | **40%** | **50%** |
| **1** | 0.089 ± 0.008 | **0.089 ± 0.008** | 0.090 ± 0.009 | 0.091 ± 0.009 | 0.091 ± 0.009 |
| **10** | 0.089 ± 0.008 | **0.089 ± 0.008** | 0.089 ± 0.008 | 0.089 ± 0.008 | 0.089 ± 0.009 |
| **100** | **0.089 ± 0.008** | 0.089 ± 0.008 | 0.089 ± 0.008 | 0.089 ± 0.008 | 0.089 ± 0.009 |
| **500** | **0.089 ± 0.008** | 0.089 ± 0.008 | 0.089 ± 0.008 | 0.089 ± 0.008 | 0.089 ± 0.009 |

| | **Ringnorm** (PAC-Bayes Test Error = 0.026±0.005) | | | | |
|---|---|---|---|---|---|
| Scalings | Percentage of training set used to compute the prior | | | | |
| | **10 %** | **20%** | **30%** | **40%** | **50%** |
| **1** | **0.025 ± 0.004** | 0.030 ± 0.007 | 0.038 ± 0.005 | 0.036 ± 0.007 | 0.038 ± 0.005 |
| **10** | **0.020 ± 0.007** | 0.021 ± 0.007 | 0.021 ± 0.007 | 0.025 ± 0.008 | 0.026 ± 0.008 |
| **100** | **0.020 ± 0.007** | 0.021 ± 0.007 | 0.021 ± 0.007 | 0.025 ± 0.008 | 0.026 ± 0.008 |
| **500** | **0.020 ± 0.007** | 0.021 ± 0.007 | 0.021 ± 0.007 | 0.025 ± 0.008 | 0.025 ± 0.005 |

Table 4: Averaged Test Error Rate corresponding to the model determined by the bound for the different settings of Table 3.

| **Problem** | **Cross-validation error rate** | **Test error rate** |
|---|---|---|
| Wdbc | 0.060 ± 0.006 | 0.072 ± 0.024 |
| Image | 0.022 ± 0.002 | 0.024 ± 0.008 |
| Waveform | 0.079 ± 0.011 | 0.085 ± 0.009 |
| Ringnorm | 0.015 ± 0.001 | 0.017 ± 0.004 |

Table 5: Averaged test error rate. For every partition we select the test error rate corresponding to the model reporting the smaller cross-validation error.

However, the comparison with Table 5 points out that the PAC-Bayes bound is not as accurate as Ten Fold cross-validation when it comes to selecting a model that yields a low test error rate. Nevertheless, in two out of the four problems (`waveform`, and `wdbc`) the bound provided a model as good as the one found by cross-validation, added to the fact that in `ringnorm` the error bars overlap. We conclude the discussion by pointing that the Cross-validation error rate cannot be used directly as a prediction on the expected test error rate in the sense of worse case performances. Of course the values of the cross-validation error rate and the test error rate are close, but it is difficult to predict how close they are going to be.

# 5   Conclusions and ongoing research

In this paper we have presented a version of the PAC-Bayes bound for linear classifiers that introduces the learning of the prior distribution over the classifiers. This prior distribution is a Gaussian with identity covariance matrix. The mean weight vector is learnt in the following way: its direction is determined from a separate subset of the training examples, while its length has to be chosen from an a priori fixed set of lengths.

The experimental work shows that this new version of the bound achieves tighter predictions of the generalization error of the stochastic classifier, compared to the original PAC-Bayes bound predictions. Moreover, if the model selection is driven by the bound, the Prior PAC-Bayes does not degrade the quality of the model selected by the original bound. Nevertheless, it has to be said that in some of our experiments the model selected by the bounds resulted as accurate as the ones selected by ten-fold cross-validation in terms of test error rate on a separate test. This fact is remarkable since to include the model selection in the training of the classifier roughly multiplies by ten the computational burden of the training when using ten-fold cross-validation but roughly by two when using the prior PAC-Bayes bound. Of course the original PAC-Bayes provides with a cheaper model selection, but its predictions about the generalization capabilities are more pessimistic.

The amount of training patterns used to learn the prior seems to be a key aspect in the goodness of this prior and thus in the tightness of the bound. Therefore, ongoing research includes methods to systematically determine an amount of patterns that provides with suitable priors. Another line of research explores the use of these bounds to reinforce different properties of the design of classifiers, such as sparsity. Finally, a deeper study about which dataset structure causes differences among the performances of cross-validation and bound-driven model selections is also being carried out.

### Acknowledgments

This work has been supported by the IST Programme of the European Community under the PASCAL Network of Excellence IST2002-506788. E. P-H. acknowledges support from Spain CICYT grant TEC2005-04264/TCM.

## Footnotes

[1]We are considering here unbiased classifiers, i.e., with $b = 0$.

# References

[1] C L Blake and C J Merz. *UCI Repository of machine learning databases*. University of California, Irvine, Dept. of Information and Computer Sciences, [http://www.ics.uci.edu/~mlearn/MLRepository.html], 1998.

[2] Bernhard E. Boser, Isabelle Guyon, and Vladimir Vapnik. A training algorithm for optimal margin classifiers. In *Computational Learing Theory*, pages 144–152, 1992.

[3] J Langford. Tutorial on practical prediction theory for classification. *Journal of Machine Learning Research*, 6(Mar):273–306, 2005.

[4] J Langford and J Shawe-Taylor. PAC-Bayes & Margins. In *Advances in Neural Information Processing Systems*, volume 14, Cambridge MA, 2002. MIT Press.

[5] D McAllester. Pac-bayesian stochastic model selection. *Machine Learning*, 51(1):5–21, 2003.

[6] M Seeger. PAC-Bayesian Generalization Error Bounds for Gaussian Process Classification. *Journal of Machine Learning Research*, 3:233–269, 2002.

[7] J Shawe-Taylor, P L Bartlett, R C Williamson, and M Anthony. Structural risk minimization over data-dependent hierarchies. *IEEE Trans. Information Theory*, 44(5):1926 – 1940, 1998.
